# Fast Information Value
# for Graphical Models

**Brigham S. Anderson**
School of Computer Science
Carnegie Mellon University
Pittsburgh, PA 15213
brigham@cmu.edu

**Andrew W. Moore**
School of Computer Science
Carnegie Mellon University
Pittsburgh, PA 15213
awm@cs.cmu.edu

## Abstract

Calculations that quantify the dependencies between variables are vital to many operations with graphical models, e.g., active learning and sensitivity analysis. Previously, pairwise information gain calculation has involved a cost quadratic in network size. In this work, we show how to perform a similar computation with cost linear in network size. The loss function that allows this is of a form amenable to computation by dynamic programming. The message-passing algorithm that results is described and empirical results demonstrate large speedups without decrease in accuracy. In the cost-sensitive domains examined, superior accuracy is achieved.

## 1 Introduction

In a diagnosis problem, one wishes to select the best test (or observation) to make in order to learn the most about a system of interest. Medical settings and disease diagnosis immediately come to mind, but sensor management (Krishnamurthy, 2002), sensitivity analysis (Kjrulff & van der Gaag, 2000), and active learning (Anderson & Moore, 2005) all make use of similar computations. These generally boil down to an all-pairs analysis between observable variables (queries) and the variables of interest (targets.)

A common technique in the field of diagnosis is to compute the mutual information between each query and target, then select the query that is expected to provide the most information (Agostak & Weiss, 1999). Likewise, a sensitivity analysis between the query variable and the target variables can be performed (Laskey, 1995; Kjrulff & van der Gaag, 2000). However, both suffer from a quadratic blowup with respect to the number of queries and targets.

In the current paper we present a loss function which can be used in a message-passing framework to perform the all-pairs computation with cost linear in network size. We describe the loss function in Section 2, we describe a polynomial expression for network-wide expected loss in Section 3, and in Section 4 we present a message-passing scheme to perform this computation efficiently for each node in the network. Section 5 shows the empirical speedups and accuracy gains achieved by the algorithm.

## 1.1 Graphical Models

To simplify presentation, we will consider only Bayesian networks, but the results generalize to any graphical model. We also restrict the class of networks to those without undirected loops, or polytrees, of which Junction trees are a member. We have a Bayesian Network $\mathcal{B}$, which is composed of an independence graph, $\mathcal{G}$ and parameters for CPT tables. The independence graph $\mathcal{G} = (\mathcal{X}, \mathcal{E})$ is a directed acyclic graph (DAG) in which $\mathcal{X}$ is a set of $N$ discrete random variables $\{x_1, x_2, ..., x_N\} \in \mathcal{X}$, and the edges define the independence relations. We will denote the marginal distribution of a single node $P(x|\mathcal{B})$ by $\pi_x$, where $(\pi_x)_i$ is $P(x = i)$. We will omit conditioning on $\mathcal{B}$ for the remainder of the paper. We indicate the number states a node $x$ can assume as $|x|$.

Additionally, each node $x$ is assigned a *cost matrix* $C_x$, in which $(C_x)_{ij}$ is the cost of believing $x = j$ when in fact the true value $x^* = i$. A cost matrix of all zeros indicates that one is not interested in the node's value. The cost matrix $C$ is useful because inhomogeneous costs are a common feature in most realistic domains. This ubiquity results from the fact that information almost always has a *purpose*, so that some variables are more relevant than others, some states of a variable are more relevant than others, and confusion between some pairs of states are more relevant than between other pairs.

For our task, we are given $\mathcal{B}$, and wish to estimate $P(\mathcal{X})$ accurately by iteratively selecting the next node to observe. Although typically only a subset of the nodes are queryable, we will for the purposes of this paper assume that any node can be queried. How do we select the most informative node to query? We must first define our objective function, which is determined by our definition of error.

## 2 Risk Due to Uncertainty

The underlying error function for the information gain computation will be denoted $Error(P(\mathcal{X})||\mathcal{X}^*)$, which quantifies the loss associated with the current belief state, $P(\mathcal{X})$ given the true values $\mathcal{X}^*$. There are several common candidates for this role, a log-loss function, a log-loss function over marginals, and an expected 0-1 misclassification rate (Kohavi & Wolpert, 1996). Constant factors have been omitted.

$$Error_{log}(P(\mathcal{X})||\mathcal{X}^*) = -\log P(\mathcal{X}^*) \tag{1}$$

$$Error_{mlog}(P(\mathcal{X})||\mathcal{X}^*) = -\sum_{u \in \mathcal{X}} \log P(u^*) \tag{2}$$

$$Error_{01}(P(\mathcal{X})||\mathcal{X}^*) = -\sum_{u \in \mathcal{X}} P(u^*) \tag{3}$$

Where $\mathcal{X}$ is the set of nodes, and $u^*$ is the true value of node $u$. The error function of Equation 1 will prove insufficient for our needs as it cannot target individual node errors, while the error function of Equation 2 results in an objective function that is quadratic in cost to compute.

We will be exploring a more general form of Equation 3 which allows arbitrary weights to be placed on different *types* of misclassifications. For instance, we would like to specify that misclassifying a node's state as 0 when it is actually 1 is different from misclassifying it as 0 when it is actually in state 2. Different costs for each node can be specified with cost matrices $C_u$ for $u \in \mathcal{X}$. The final error function is

$$Error(P(\mathcal{X})||\mathcal{X}^*) = \sum_{u \in \mathcal{X}} \sum_{i}^{|u|} P(u = i) C_u[u^*, i] \tag{4}$$

Where $C[i,j]$ is the $ij$th element of the matrix $C$, and $|u|$ is the number of states that the node $u$ can assume. The presence of the cost matrix $C_u$ in Equation 4 constitutes a significant advantage in real applications, as they often need to specify inhomogeneous costs.

There is a separate consideration, that of *query cost*, or $cost(x)$, which is the cost incurred by the action of observing $x$ (e.g., the cost of a medical test.) If both the query cost and the misclassification cost $C$ are formulated in the same units, e.g., dollars, then they form a coherent decision framework. The query costs will be omitted from this presentation for clarity.

In general, one does not actually know the true values $\mathcal{X}^*$ of the nodes, so one cannot directly minimize the error function as described. Instead, the expected error, or risk, is used.

$$Risk(P(\mathcal{X})) = \sum_{\mathbf{x}} P(\mathbf{x}) Error P(\mathcal{X}||\mathbf{x}) \tag{5}$$

which for the error function of Equation 4 reduces to

$$Risk(P(\mathcal{X})) = \sum_{u \in \mathcal{X}} \sum_j \sum_k P(u = j) P(u = k) C_u[j,k] \tag{6}$$

$$= \sum_{u \in \mathcal{X}} \pi_u^T C_u \pi_u \tag{7}$$

where $(\pi_u)_i = P(u = i)$. This is the objective we will minimize. It quantifies "On average, how much is our current ignorance going cost us?" For comparison, note that the log-loss function, $Error_{log}$, results in an entropy risk function $Risk_{log}(P(\mathcal{X})) = H(\mathcal{X})$, and the log-loss function over the marginals, $Error_{mlog}$, results in the risk function $Risk_{mlog}(P(\mathcal{X})) = \sum_{u \in \mathcal{X}} H(u)$.

Ultimately, we want to find the nodes that have the greatest effect on $Risk(P(\mathcal{X}))$, so we must condition $Risk(P(\mathcal{X}))$ on the beliefs at each node. In other words, if we learned that the true marginal probabilities of node $x$ were $\pi_x$, what effect would that have on our current risk, or rather, what is $Risk(P(\mathcal{X})|P(x) = \pi_x)$? Discouragingly, however, any change in $\pi_x$ propagates to all the other beliefs in the network. It seems as if we must perform several network evaluations for each node, a prohibitive cost for networks of any appreciable size. However, we will show that in fact dynamic programming can perform this computation for all nodes in only two passes through the network.

## 3   Risk Calculation

To clarify our objective, we wish to construct a function $R_a(\pi)$ for each node $a$, where $R_a(\pi) = Risk(P(\mathcal{X})|P(a) = \pi)$. Suppose, for instance, that we learn that the value of node $a$ is equal to 3. Our $P(\mathcal{X})$ is now constrained to have the marginal $P(a) = \pi_a'$, where $(\pi_a')_3 = 1$ and equals zero elsewhere. If we had the function $R_a$ in hand, we could simply evaluate $R_a(\pi_a')$ to immediately compute our new network-wide risk, which would account for all the changes in beliefs to all the other nodes due to learning that $a = 3$. This is exactly our objective; we would like to precompute $R_a$ for all $a \in \mathcal{X}$. Define

$$R_a(\pi) = Risk(P(\mathcal{X})|P(a) = \pi) \tag{8}$$

$$= \sum_{u \in \mathcal{X}} \pi_u^T C_u \pi_u \bigg|_{P(a)=\pi} \tag{9}$$

This simply restates the risk definition of Equation 7 under the condition that $P(a) = \pi$. As shown in the next theorem, the function $R_a$ has a surprisingly simple form.

**Theorem 3.1.** *For any node $x$, the function $R_x(\pi)$ is a second-degree polynomial function of the elements of $\pi$*

*Proof.* Define the matrix $\mathbf{P}_{u|v}$ for every pair of nodes $(u, v)$, such that $(\mathbf{P}_{u|v})_{ij} = P(u = j|v = i)$. Recall that the the beliefs at node $x$ have a strictly linear relationship to the beliefs of node $u$, since

$$(\pi_u)_i = \sum_k P(u = i|x = k)P(x = k) \tag{10}$$

is equivalent to $\pi_u = \mathbf{P}_{u|x}\pi_x$. Substituting $\mathbf{P}_{u|x}\pi_x$ for $\pi_u$ in Equation 9 obtains

$$R_x(\pi) = \sum_{u \in \mathcal{X}} \pi_x^T \mathbf{P}_{u|x}^T C_u \mathbf{P}_{u|x}\pi_x \bigg|_{\pi_x = \pi} \tag{11}$$

$$= \pi^T \left( \sum_{u \in \mathcal{X}} \mathbf{P}_{u|x}^T C_u \mathbf{P}_{u|x} \right) \pi \tag{12}$$

$$= \pi^T \Theta_x \pi \tag{13}$$

Where $\Theta_x$ is an $|x| \times |x|$ matrix.

$\square$

Note that the matrix $\Theta_x$ is sufficient to completely describe $R_x$, so we only need to consider the computation of $\Theta_x$ for $x \in \mathcal{X}$. From Equation 12, we see a simple equation for computing these $\Theta_x$ directly (though expensively):

$$\Theta_x = \sum_{u \in \mathcal{X}} \mathbf{P}_{u|x}^T C_u \mathbf{P}_{u|x} \tag{14}$$

**Example #1**
Given the 2-node network $a \to b$, how do we calculate $R_a(\pi)$, the total risk associated with our beliefs about the value of node $a$? Our objective is thus to determine

$$R_a(\pi) = Risk(P(a, b)|P(a) = \pi) \tag{15}$$

$$= \pi^T \Theta_a \pi \tag{16}$$

Equation 14 will give $\Theta_a$ as

$$\Theta_a = \mathbf{P}_{a|a}^T C_a \mathbf{P}_{a|a} + \mathbf{P}_{b|a}^T C_b \mathbf{P}_{b|a} \tag{17}$$

$$= C_a + \mathbf{P}_{b|a}^T C_b \mathbf{P}_{b|a} \tag{18}$$

with $\mathbf{P}_{a|a} = \mathbf{I}$ by definition. The individual coefficients of $\Theta_a$ are thus

$$\theta_{aij} = C_{aij} + \sum_k \sum_l P(b = k|a = i)P(b = l|a = j)C_{bkl} \tag{19}$$

Now we can compute the relation between any marginal $\pi$ at node $a$ and our total network-wide risk via $R_a(\pi)$. However, using Equation 14 to compute all the $\Theta$ would require evaluating the entire network once per node. The function can, however, be decomposed further, which will enable much more efficient computation of $\Theta_x$ for $x \in \mathcal{X}$.

## 3.1 Recursion

To create an efficient message-passing algorithm for computing $\Theta_x$ for all $x \in \mathcal{X}$, we will introduce $\Theta_x^{\mathcal{W}}$, where $\mathcal{W}$ is a *subset* of the network over which $Risk(P(\mathcal{X}))$ is summed.

$$\Theta_x^{\mathcal{W}} = \sum_{u \in \mathcal{W}} \mathbf{P}_{u|x}^T C_u \mathbf{P}_{u|x} \tag{20}$$

This is otherwise identical to Equation 14. It implies, for instance, that $\Theta_x^x = C_x$. More importantly, these matrices can be usefully decomposed as follows.

**Theorem 3.2.** $\Theta_x^{\mathcal{W}} = \mathbf{P}_{y|x}^T \Theta_y^{\mathcal{W}} \mathbf{P}_{y|x}$   *if $x$ and $\mathcal{W}$ are conditionally independent given $y$.*

*Proof.* Note that $\mathbf{P}_{u|x} = \mathbf{P}_{u|y}\mathbf{P}_{y|x}$ for $u \in \mathcal{X}$, since

$$(\mathbf{P}_{u|y}\mathbf{P}_{y|x})_{ij} = \sum_k^{|y|} P(u = i|y = k)P(y = k|x = j) \tag{21}$$

$$= P(u = i|x = j) \tag{22}$$

$$= (\mathbf{P}_{u|x})_{ij} \tag{23}$$

Step (21) is only true if $x$ and $u$ are conditionally independent given $y$. Substituting this result into Equation 20, we conclude

$$\Theta_x^{\mathcal{W}} = \sum_{u \in \mathcal{W}} \mathbf{P}_{u|x}^T C_u \mathbf{P}_{u|x} \tag{24}$$

$$= \sum_{u \in \mathcal{W}} \mathbf{P}_{y|x}^T \mathbf{P}_{u|y}^T \Theta_u^u \mathbf{P}_{u|y} \mathbf{P}_{y|x} \tag{25}$$

$$= \mathbf{P}_{y|x}^T \left( \sum_{u \in \mathcal{W}} \mathbf{P}_{u|y}^T \Theta_u^u \mathbf{P}_{u|y} \right) \mathbf{P}_{y|x} \tag{26}$$

$$= \mathbf{P}_{y|x}^T \Theta_y^{\mathcal{W}} \mathbf{P}_{y|x} \tag{27}$$

$\square$

### Example #2

Suppose we now have a 3-node network, $a \to b \to c$, and we are only interested in the effect that node $a$ has on the network-wide $Risk$. Our objective is to compute

$$R_a(\pi) = Risk(P(a, b, c)|P(a) = \pi) \tag{28}$$

$$= \pi^T \Theta_a \pi \tag{29}$$

where $\Theta_a$ is by definition

$$\Theta_a = \Theta_a^{abc} \tag{30}$$

$$= C_a + \mathbf{P}_{b|a}^T C_b \mathbf{P}_{b|a} + \mathbf{P}_{c|a}^T C_c \mathbf{P}_{c|a} \tag{31}$$

Using Theorem 3.2 and the fact that $a$ is conditionally independent of $c$ given $b$, we know

$$\Theta_a^{abc} = \Theta_a^a + \mathbf{P}_{b|a}^T \Theta_b^{bc} \mathbf{P}_{b|a} \tag{32}$$

$$\Theta_b^{bc} = \Theta_b^b + \mathbf{P}_{c|b}^T \Theta_c^c \mathbf{P}_{c|b} \tag{33}$$

Substituting 33 into 32

$$\Theta_a = \Theta_a^a + \mathbf{P}_{b|a}^T \left( \Theta_b^b + \mathbf{P}_{c|b}^T \Theta_c^c \mathbf{P}_{c|b} \right) \mathbf{P}_{b|a} \tag{34}$$

$$= C_a + \mathbf{P}_{b|a}^T \left( C_b + \mathbf{P}_{c|b}^T C_c \mathbf{P}_{c|b} \right) \mathbf{P}_{b|a} \tag{35}$$

Note that the coefficient $\Theta_a$ is obtained from probabilities between neighboring nodes only, without having to explicitly compute $\mathbf{P}_{c|a}$.

## 4 Message Passing

We are now ready to define message passing. Messages are of two types; *in*-messages and *out*-messages. They are denoted by $\lambda$ and $\mu$, respectively. Out-messages $\mu$ are passed from parent to child, and in-messages $\lambda$ are passed from child to parent. The messages from $x$ to $y$ will be denoted as $\mu_{xy}$ and $\lambda_{xy}$. In the discrete case, $\mu_{xy}$ and $\lambda_{xy}$ will both be matrices of size $|y| \times |y|$. The messages summarize the effect that $y$ has on the part of the network that $y$ is d-separated from by $x$. Messages relate to the $\Theta$ coefficients by the following definition

$$\lambda_{yx} = \Theta_x^{\backslash y} \tag{36}$$

$$\mu_{yx} = \Theta_x^{\backslash y} \tag{37}$$

where the (nonstandard) notation $\Theta_x^{\backslash y}$ indicates the matrix $\Theta_x^{\mathcal{V}}$ for which $\mathcal{V}$ is the set of all the nodes in $\mathcal{X}$ that are reachable by $x$ if $y$ were removed from the graph. In other words, $\Theta_x^{\backslash y}$ is summarizing the effect that $x$ has on the entire network *except* for the part of the network that $x$ can only reach through $y$.

**Propagation:** The message-passing scheme is organized to recursively compute the $\Theta$ matrices using Theorem 3.2. As can be seen from Equations 36 and 37, the two types of messages are very similar in meaning. They differ only in that passing a message from a parent to child automatically separates the child from the rest of the network the parent is connected to, while a child-to-parent message does not necessarily separate the parent from the rest of the network that the child is connected to (due to the "explaining away" effect.)

The contruction of the $\mu$-message involves a short sequence of basic linear algebra. The $\mu$-message from $x$ to child $c$ is created from all other messages entering $x$ except those from $c$. The definition is

$$\mu_{xc} = \mathbf{P}_{x|c}^T \left( C_x + \sum_{u \in pa(x)} \mu_{ux} + \sum_{v \in ch(x) \backslash c} \lambda_{vx} \right) \mathbf{P}_{x|c} \tag{38}$$

The $\lambda$-messages from $x$ to parent $u$ are only slightly more involved. To account for the "explaining away" effect, we must construct $\lambda_{xu}$ directly from the parents of $x$.

$$\lambda_{xu} = \mathbf{P}_{x|u}^T \left( C_x + \sum_{c \in ch(x)} \lambda_{cx} \right) \mathbf{P}_{x|u} +$$

$$\sum_{w \in pa(x) \backslash u} \mathbf{P}_{w|u}^T \left( C_w + \sum_{v \in pa(w)} \mu_{vw} + \sum_{c \in ch(w) \backslash x} \lambda_{cw} \right) \mathbf{P}_{w|u} \tag{39}$$

Messages are constructed (or "sent") whenever all of the required incoming messages are present *and* that particular message has not already been sent. For example, the out-message $\mu_{xc}$ can be sent only when messages from all the parents of $x$ and all the children of $x$ (save $c$) are present. The overall effect of this constraint is a single leaves-inward propagation followed by a single root-outward propagation.

**Initialization and Termination:** Initialization occurs naturally at any singly-connected (leaf) node $x$, where the message is by definition $C_x$. Termination occurs when no more messages meet the criteria for sending. Once all message propagation is finished, for each node $x$ the coefficients $\Theta_x$ can be computed by a simple summation:

$$\Theta_x = \sum_{c \in ch(x)} \lambda_{cx} + \sum_{u \in par(x)} \mu_{ux} + C_x \tag{40}$$

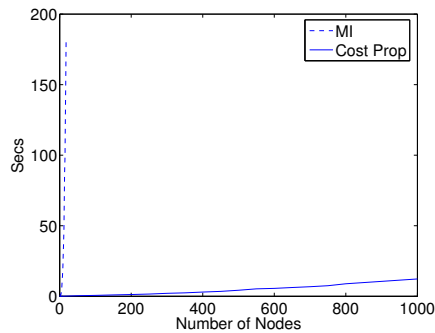

Figure 1: Comparison of execution times with synthetic polytrees.

Propagation runs in time linear in the number of nodes once the initial local probabilities are calculated. The local probabilities required are the matrices $\mathbf{P}$ for each parent-child probability, e.g., $P(child = j|parent = i)$, and for each *pair* (not set) of parents that share a child, $P(parent = j|coparent = i)$. These are all immediately available from a junction tree, or they can be obtained with a run of belief propagation.

It is worth noting that the apparent complexity of the $\lambda, \mu$ message propagation equations is due to the Bayes Net representation. The equivalent factor graph equations (not shown) are markedly more succinct.

## 5  Experiments

The performance of the message-passing algorithm (hereafter CostProp) was compared with a standard information gain algorithm which uses mutual information (hereafter MI). The error function used by MI is from Equation 2, where $Error_{mlog}(P(\mathcal{X})||\mathcal{X}^*) = \sum_{x \in \mathcal{X}} \log P(x^*)$, with a corresponding risk function $Risk(P(\mathcal{X})) = \sum_{x \in \mathcal{X}} H(x)$. This corresponds to selecting the node $x$ that has the highest summed mutual information with each of the target nodes (in this case, the set of target nodes is $\mathcal{X}$ and the set of query nodes is also $\mathcal{X}$.) The computational cost of MI grows quadratically as the product of the number of queries and of targets.

In order to test the speed and relative accuracy of CostProp, we generated random polytrees with varying numbers of trinary nodes. The CPT tables were randomly generated with a slight bias towards lower-entropy probabilities. The code was written in Matlab using the Bayes Net Toolbox (Murphy, 2005).

**Speed:** We generated polytrees of sizes ranging from 2 to 1000 nodes and ran the MI algorithm, the CostProp algorithm, and a random-query algorithm on each. The two non-random algorithms were run using a junction tree, the build time of which was not included in the reported run times of either algorithm. Even with the relatively slow Matlab code, the speedup shown in Figure 1 is obvious. As expected, CostProp is many orders of magnitude faster than the MI algorithm, and shows a qualitative difference in scaling properties.

**Accuracy:** Due to the slow running time of MI, the accuracy comparison was performed on polytrees of size 20. For each run, a true assignment $\mathcal{X}^*$ was generated from the tree, but were initially hidden from the algorithms. Each algorithm would then determine for itself the best node to observe, receive the true value of that node, then select the next node to observe, et cetera. The true error at each step was computed as the 0-1 error of Equation 3. The reduction in error plotted against number of queries is shown in Figure 2. With uniform

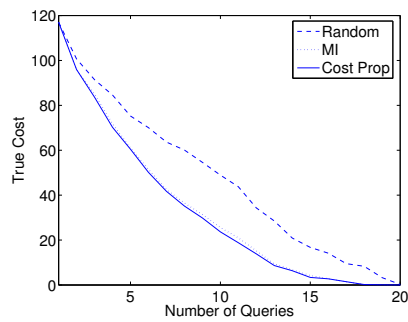
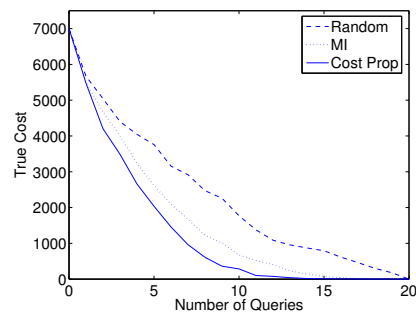

Figure 2: Performance on synthetic poly-trees with symmetric costs.

Figure 3: Performance on synthetic poly-trees with asymmetric costs.

cost matrices, performance of MI and CostProp are approximately equal on this task, but both are better than random. We next made the cost matrices asymmetric by initializing them such that confusing one pair of states was 100 times more costly than confusing the other two pairs. The results of Figure 3 show that CostProp reduces error faster than MI, presumably because it can accomodate the cost matrix information.

# 6 Discussion

We have described an all-pairs information gain calculation that scales linearly with network size. The objective function used has a polynomial form that allows for an efficient message-passing algorithm. Empirical results demonstrate large speedups and even improved accuracy in cost-sensitive domains. Future work will explore other applications of this method, including sensitivity analysis and active learning. Further research into other uses for the belief polynomials will also be explored.

# References

Agostak, J. M., & Weiss, J. (1999). Active Fusion for Diagnosis Guided by Mutual Information. *Proceedings of the 2nd International Conference on Information Fusion.*

Anderson, B. S., & Moore, A. W. (2005). Active learning for hidden markov models: Objective functions and algorithms. *Proceedings of the 22nd International Conference on Machine Learning.*

Kjrulff, U., & van der Gaag, L. (2000). Making sensitivity analysis computationally efficient.

Kohavi, R., & Wolpert, D. H. (1996). Bias Plus Variance Decomposition for Zero-One Loss Functions. *Machine Learning : Proceedings of the Thirteenth International Conference.* Morgan Kaufmann.

Krishnamurthy, V. (2002). Algorithms for optimal scheduling and management of hidden markov model sensors. *IEEE Transactions on Signal Processing*, *50*, 1382–1397.

Laskey, K. B. (1995). Sensitivity Analysis for Probability Assessments in Bayesian Networks. *IEEE Transactions on Systems, Man, and Cybernetics.*

Murphy, K. (2005). *Bayes net toolbox for matlab*. U. C. Berkeley. `http://www.ai.mit.edu/~murphyk/Software/BNT/bnt.html`.
